# Global Ranking Using Continuous Conditional Random Fields

[1]**Tao Qin**, [1]**Tie-Yan Liu**, [2]**Xu-Dong Zhang**, [2]**De-Sheng Wang**, [1]**Hang Li**
[1]Microsoft Research Asia, [2]Tsinghua University
[1]{taoqin, tyliu, hangli}@microsoft.com
[2]{zhangxd, wangdsh_ee}@tsinghua.edu.cn

## Abstract

This paper studies global ranking problem by learning to rank methods. Conventional learning to rank methods are usually designed for 'local ranking', in the sense that the ranking model is defined on a single object, for example, a document in information retrieval. For many applications, this is a very loose approximation. Relations always exist between objects and it is better to define the ranking model as a function on all the objects to be ranked (i.e., the relations are also included). This paper refers to the problem as global ranking and proposes employing a Continuous Conditional Random Fields (CRF) for conducting the learning task. The Continuous CRF model is defined as a conditional probability distribution over ranking scores of objects conditioned on the objects. It can naturally represent the content information of objects as well as the relation information between objects, necessary for global ranking. Taking two specific information retrieval tasks as examples, the paper shows how the Continuous CRF method can perform global ranking better than baselines.

## 1 Introduction

Learning to rank is aimed at constructing a model for ordering objects by means of machine learning. It is useful in many areas including information retrieval, data mining, natural language processing, bioinformatics, and speech recognition. In this paper, we take information retrieval as an example.

Traditionally learning to rank is restricted to 'local ranking', in which the ranking model is defined on a single object. In other words, the relations between the objects are not directly represented in the model. In many application tasks this is far from being enough, however. For example, in Pseudo Relevance Feedback [17, 8], we manage to rank documents on the basis of not only relevance of documents to the query, but also similarity between documents. Therefore, the use of a model solely based on individual documents would not be sufficient. (Previously, heuristic methods were developed for Pseudo Relevance Feedback.) Similar things happen in the tasks of Topic Distillation [12, 11] and Subtopic Retrieval [18]. Ideally, in information retrieval we would exploit a ranking model defined as a function on all the documents with respect to the query. In other words, ranking should be conducted on the basis of the contents of objects as well as the relations between objects. We refer to this setting as 'global ranking' and give a formal description on it with information retrieval as an example.

Conditional Random Fields (CRF) technique is a powerful tool for relational learning, because it allows the uses of both relations between objects and contents of objects [16]. However, conventional CRF cannot be directly applied to global ranking because it is a discrete model in the sense that the output variables are discrete [16]. In this work, we propose a *Continuous* CRF model (C-CRF) to deal with the problem. The C-CRF model is defined as a conditional probability distribution over ranking scores of objects (documents) conditioned on the objects (documents). The specific

probability distribution can be represented by an undirected graph, and the output variables (ranking scores) can be continuous. To our knowledge, this is the first time such kind of CRF model is proposed.

We apply C-CRF to two global ranking tasks: Pseudo Relevance Feedback and Topic Distillation. Experimental results on benchmark data show that our method performs better than baseline methods.

## 2 Global Ranking Problem

Document ranking in information retrieval is a problem as follows. When the user submits a query, the system retrieves all the documents containing at least one query term, calculates a ranking score for each of the documents using the ranking model, and sorts the documents according to the ranking scores. The scores can represent relevance, importance, and/or diversity of documents.

Let $q$ denote a query. Let $x^{(q)} = \{x_1^{(q)}, x_2^{(q)}, \ldots, x_{n^{(q)}}^{(q)}\}$ denote the documents retrieved with $q$, and $y^{(q)} = \{y_1^{(q)}, y_2^{(q)}, \ldots, y_{n^{(q)}}^{(q)}\}$ denote the ranking scores assigned to the documents. Here $n^{(q)}$ stands for the number of documents retrieved with $q$. Note that the numbers vary according to queries. We assume that $y^{(q)}$ is determined by a ranking model.

We call the ranking 'local ranking', if the ranking model is defined as

$$y_i^{(q)} = f(x_i^{(q)}), i = 1, \ldots, n^{(q)} \tag{1}$$

Furthermore, we call the ranking 'global ranking', if the ranking model is defined as

$$y^{(q)} = F(x^{(q)}) \tag{2}$$

The major difference between the two is that $F$ takes on all the documents together as its input, while $f$ takes on an individual document as its input. In other words, in global ranking, we use not only the content information of documents but also the relation information between documents. There are many specific application tasks that can be viewed as examples of global ranking. These include Pseudo Relevance Feedback, Topic Distillation, and Subtopic Retrieval.

## 3 Continuous CRF for Global Ranking

### 3.1 Continuous CRF

Let $\{h_k(y_i^{(q)}, x^{(q)})\}_{k=1}^{K_1}$ be a set of real-valued feature functions defined on document set $x^{(q)}$ and ranking score $y_i^{(q)}$ $(i = 1, \cdots, n^{(q)})$, and $\{g_k(y_i^{(q)}, y_j^{(q)}, x^{(q)})\}_{k=1}^{K_2}$ be a set of real-valued feature functions defined on $y_i^{(q)}$, $y_j^{(q)}$, and $x^{(q)}$ $(i, j = 1, \cdots, n^{(q)}, i \neq j)$.

Continuous Conditional Random Fields is a conditional probability distribution with the following density function,

$$\Pr(y^{(q)}|x^{(q)}) = \frac{1}{Z(x^{(q)})} \exp\left\{\sum_i \sum_{k=1}^{K_1} \alpha_k h_k(y_i^{(q)}, x^{(q)}) + \sum_{i,j} \sum_{k=1}^{K_2} \beta_k g_k(y_i^{(q)}, y_j^{(q)}, x^{(q)})\right\}, \tag{3}$$

where $\alpha$ is a $K_1$-dimensional parameter vector and $\beta$ is a $K_2$-dimensional parameter vector, and $Z(x^{(q)})$ is a normalization function,

$$Z(x^{(q)}) = \int_{y^{(q)}} \exp\left\{\sum_i \sum_{k=1}^{K_1} \alpha_k h_k(y_i^{(q)}, x^{(q)}) + \sum_{i,j} \sum_{k=1}^{K_2} \beta_k g_k(y_i^{(q)}, y_j^{(q)}, x^{(q)})\right\} dy^{(q)}. \tag{4}$$

Given a set of documents $x^{(q)}$ for a query, we select the ranking score vector $y^{(q)}$ with the maximum conditional probability $\Pr(y^{(q)}|x^{(q)})$ as the output of our proposed global ranking model:

$$F(x^{(q)}) = \arg\max_{y^{(q)}} \Pr(y^{(q)}|x^{(q)}). \tag{5}$$

C-CRF is a graphical model, as depicted in Figure 1. In the conditioned undirected graph, a white vertex represents a ranking score, a gray vertex represents a document, an edge between two white vertexes represents the dependency between ranking scores, and an edge between a gray vertex and a white vertex represents the dependency of a ranking score on its document (content). (In principle a ranking score can depend on all the documents of the query; here for ease of presentation we only consider the simple case in which it only depends on the corresponding document.)

In C-CRF, feature function $h_k$ represents the dependency between the ranking score of a document and the content of it, and feature function $g_k$ represents a relation between the ranking scores of two documents. Different retrieval tasks may have different relations (e.g. similarity relation, parent-child relation), as will be explained in Section 4. For ease of reference, we call the feature functions $h_k$ vertex features, and the feature functions $g_k$ edge features.

Note that in conventional CRF the output random variables are discrete while in C-CRF the output variables are continuous. This makes the inference of C-CRF largely different from that of conventional CRF, as will be seen in Section 4.

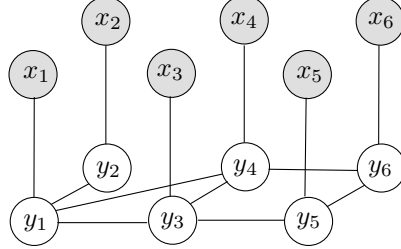

Figure 1: Continuous CRF Model

## 3.2 Learning

In the inference of C-CRF, the paramters $\{\alpha, \beta\}$ are given, while in learning, they are to be estimated.

Given training data $\{x^{(q)}, y^{(q)}\}_{q=1}^{N}$, where each $x^{(q)} = \{x_1^{(q)}, x_2^{(q)}, ..., x_{n^{(q)}}^{(q)}\}$ is a set of documents of query $q$, and each $y^{(q)} = \{y_1^{(q)}, y_2^{(q)}, ..., y_{n^{(q)}}^{(q)}\}$ is a set of ranking scores associated with the documents of query $q$, we employ Maximum Likelihood Estimation to estimate the parameters $\{\alpha, \beta\}$ of C-CRF. Specifically, we calculate the conditional log likelihood of the training data with respect to the C-CRF model,

$$L(\alpha, \beta) = \sum_{q=1}^{N} \log \Pr(y^{(q)}|x^{(q)}; \alpha, \beta). \tag{6}$$

We then use Gradient Ascend to maximze the log likelihood, and use the optimal parameter $\hat{\alpha}, \hat{\beta}$ to rank the documents of a new query.

# 4 Case Study

## 4.1 Pseudo Relevance Feedback (PRF)

Pseudo Relevance Feedback (PRF) [17, 8] is an example of global ranking, in which similarity between documents are considered in the ranking process. Conceptually, in this task one first conducts a round of ranking, assuming that the top ranked documents are relevant; then conducts another round of ranking, using similarity information between the top ranked documents and the other documents to boost some relevant documents dropped in the first round. The underlying assumption is that *similar documents are likely to have similar ranking scores*. Here we consider a method of using C-CRF for performing the task.

### 4.1.1 Continuous CRF for Pseudo Relevance Feedback

We first introduce vertex feature functions. The relevance of a document to the query depends on many factors, such as term frequency, page importance, and so on. For each factor we define a vertex feature function. Suppose that $x_{i,k}^{(q)}$ is the $k$-th relevance factor of document $x_i$ with respect to query

$q$ extracted by operator $t_k$: $x_{i,k}^{(q)} = t_k(x_i, q)$. We define the $k$-th feature function[1] $h_k(y_i, x)$ as

$$h_k(y_i, x) = -(y_i - x_{i,k})^2. \tag{7}$$

Next, we introduce the edge feature function. Recall that there are two rounds in PRF: the first round scores each document, and the second round re-ranks the documents considering similarity between documents. Here the similarities between any two documents are supposed to be given. We incorporate them into the edge feature function.

$$g(y_i, y_j, x) = -\frac{1}{2} S_{i,j}(y_i - y_j)^2, \tag{8}$$

where $S_{i,j}$ is similarity between documents $x_i$ and $x_j$, which can be extracted by some operator $s$ from the raw content[2] of document $x_i$ and $x_j$: $S_{i,j} = s(x_i, x_j)$. The larger $S_{i,j}$ is, the more similar the two documents are. Sine only similarity relation is considered in this task, we have only one edge function ($K_2 = 1$).

The C-CRF for Pseudo Relevance Feedback then becomes

$$\Pr(y|x) = \frac{1}{Z(x)} \exp \left\{ \sum_i \sum_{k=1}^{K_1} -\alpha_k(y_i - x_{i,k})^2 + \sum_{i,j} -\frac{\beta}{2} S_{i,j}(y_i - y_j)^2 \right\}, \tag{9}$$

where $Z(x)$ is defined as

$$Z(x) = \int_y \exp \left\{ \sum_i \sum_{k=1}^{K_1} -\alpha_k(y_i - x_{i,k})^2 + \sum_{i,j} -\frac{\beta}{2} S_{i,j}(y_i - y_j)^2 \right\} dy. \tag{10}$$

To guarantee that $\exp \left\{ \sum_i \sum_{k=1}^{K_1} -\alpha_k(y_i - x_{i,k})^2 + \sum_{i,j} -\frac{\beta}{2} S_{i,j}(y_i - y_j)^2 \right\}$ is integrable, we must have $\alpha_k > 0$[3] and $\beta > 0$.

The item $\sum_i \sum_{k=1}^{K_1} -\alpha_k(y_i - x_{i,k})^2$ in Eq. (9) plays a role similar to the first round of PRF: the ranking score $y_i$ is determined solely by the relevance factors of document $x_i$. The item $\sum_{i,j} -\frac{\beta}{2} S_{i,j}(y_i - y_j)^2$ in Eq. (9) plays a role similar to the second round of PRF: it makes sure that similar documents have similar ranking scores. We can see that CRF combines the two rounds of ranking of PRF into one.

To rank the documents of a query, we calculate the ranking scores of documents with respect to this query in the following way.

$$F(x) = \arg\max_y \Pr(y|x; \alpha, \beta) = (\alpha^T eI + \beta D - \beta S)^{-1} X\alpha. \tag{11}$$

where $e$ is a $K_1$-dimensional all-ones vector, $I$ is an $n \times n$ identity matrix, $S$ is a similarity matrix with $S_{i,j} = s(x_i, x_j)$, $D$ is an $n \times n$ diagonal matrix with $D_{i,i} = \sum_j S_{i,j}$, and $X$ is a factor matrix with $X_{i,k} = x_{i,k}$. If we ignore the relation between documents and set $\beta = 0$, then the ranking model degenerates to $F(x) = X\alpha$, which is equivalent to a linear model used in conventional local ranking.

For $n$ documents, the time complexity of straightforwardly computing the ranking model (11) is of order $O(n^3)$ and thus the computation is expensive. The main cost of the computation comes from matrix inversion. We employ a fast computation technique to quickly perform the task. First, we make $S$ a sparse matrix, which has at most $K$ non-zero values in each row and each column. We can do so by only considering the similarity between each document and its $\frac{K}{2}$ nearest neighbors. Next, we use the Gibbs-Poole-Stockmeyer algorithm [9] to convert $S$ to a banded matrix. Finally we solve the following system of linear equation and take the solution as ranking scores.

$$(\alpha^T eI + \beta D - \beta S)F(x) = X\alpha \tag{12}$$

Since $S$ is a banded matrix, the scores $F(x)$ in Eq.(12) can be computed with time complexity of $O(n)$ when $K \ll n$ [5]. That is to say, the time complexity of testing a new query is comparable with those of existing local ranking methods.

**Algorithm 1** Learning Algorithm of Continuous CRF for Pseudo Relevance Feedback

---

**Input:** training data $\{(x^{(1)}, y^{(1)}), \cdots, (x^{(N)}, y^{(N)})\}$, number of iterations $T$ and learning rate $\eta$
Initialize parameter $\log \alpha_k$ and $\log \beta$
**for** $t = 1$ **to** $T$ **do**
   **for** $i = 1$ **to** $N$ **do**
      Compute gradient $\nabla_{\log \alpha_k}$ and $\nabla_{\log \beta}$ using Eq. (13) and (14) for a single query $(x^{(i)}, y^{(i)}, S^{(i)})$.
      Update $\log \alpha_k = \log \alpha_k + \eta \times \nabla_{\log \alpha_k}$ and $\log \beta = \log \beta + \eta \times \nabla_{\log \beta}$
   **end for**
**end for**
**Output:** parameters of CRF model $\alpha_k$ and $\beta$.

---

### 4.1.2 Learning

In learning, we try to maximize the log likelihood. Note that maximization of $L(\alpha, \beta)$ in Eq. (6) is a constrained optimization problem because we need to guarantee that $\alpha_k > 0$ and $\beta > 0$. Gradient Ascent cannot be directly applied to such a constrained optimization problem. Here we adopt a technique similar to that in [3]. Specifically, we maximize $L(\alpha, \beta)$ with respect to $\log \alpha_k$ and $\log \beta$ instead of $\alpha_k$ and $\beta$. As a result, the new optimization issue becomes unconstrained and Gradient Ascent method can be used. Algorithm 1 shows the learning algorithm based on Stochastic Gradient Ascent [4], in which the gradient $\nabla_{\log \alpha_k}$ and $\nabla_{\log \beta}$ can be computed as follows[5].

$$\nabla_{\log \alpha_k} = \frac{\partial L(\alpha, \beta)}{\partial \log \alpha_k} = -\alpha_k \left\{ \left( -\frac{1}{2}(A^{-T}) :^T I : \right) + 2 X_{,k}^T A^{-1} b - b^T A^{-1} A^{-1} b + \sum_i (y_i^2 - 2 y_i x_{i,k}) \right\}$$
(13)

$$\nabla_{\log \beta} = \frac{\partial L(\alpha, \beta)}{\partial \log \beta} = -\beta \left\{ \left( -\frac{1}{2}(A^{-T}) :^T (D - S) : \right) - b^T A^{-1}(D - S) A^{-1} b + \sum_{i,j} \frac{1}{2} S_{i,j}(y_i - y_j)^2 \right\}$$
(14)

where $A = \alpha^T e I + \beta D - \beta S$, $|A|$ is determinant of matrix $A$, $b = X\alpha$, $c = \sum_i \sum_{k=1}^{K_1} \alpha_k x_{i,k}^2$, $X :$ denotes the long column vector formed by concatenating the columns of matrix $X$, and $X_{,k}$ denotes the $k$-th column of matrix $X$.

### 4.2 Topic Distillation (TD)

Topic Distillation [12] is another example of global ranking. In this task, one selects a page that can best represent the topic of the query from a web site by using structure (relation) information of the site. If both a page and its parent page are concerned with the topic, then the parent page is preferred (to be ranked higher) [12, 11]. Here we apply C-CRF to Topic Distillation.

### 4.2.1 Continuous CRF for Topic Distillation

We define the vertex feature function $h_k(y_i, x)$ in the same way as in Eq.(7).

Recall that in Topic Distillation, a page is more preferred than its child page if both of them are relevant to a query. Here the parent-child relation between two pages is supposed to be given. We incorporate them into the edge feature function. Specifically, we define the (and the only) edge feature function as

$$g(y_i, y_j, x) = R_{i,j}(y_i - y_j),$$
(15)

where $R_{i,j} = r(x_i, x_j)$ denotes the parent-child relation: $r(x_i, x_j) = 1$ if document $x_i$ is the parent of $x_j$, and $r(x_i, x_j) = 0$ for other cases.

The C-CRF for Topic Distillation then becomes

$$\Pr(y|x) = \frac{1}{Z(x)} \exp \left\{ \sum_i \sum_{k=1}^{K_1} -\alpha_k (y_i - x_{i,k})^2 + \sum_{i,j} \beta R_{i,j}(y_i - y_j) \right\},$$
(16)

where $Z(x)$ is defined as

$$Z(x) = \int_y \exp \left\{ \sum_i \sum_{k=1}^{K_1} -\alpha_k (y_i - x_{i,k})^2 + \sum_{i,j} \beta R_{i,j} (y_i - y_j) \right\} dy. \tag{17}$$

To guarantee that $\exp \left\{ \sum_i \sum_{k=1}^{K_1} -\alpha_k (y_i - x_{i,k})^2 + \sum_{i,j} \beta R_{i,j} (y_i - y_j) \right\}$ is integrable, we must have $\alpha_k > 0$.

The C-CRF can naturally model Topic Distillation: if the value of $R_{i,j}$ is one, then the value of $y_i$ is large than that of $y_j$ with high probability.

To rank the documents of a query, we calculate the ranking scores in the following way.

$$F(x) = \arg \max_y \Pr(y|x; \alpha, \beta) = \frac{1}{\alpha^T e} \left( 2X\alpha + \beta(D_r - D_c)e \right) \tag{18}$$

where $D_r$ and $D_c$ are two diagonal matrixes with $D_{ri,i} = \sum_j R_{i,j}$ and $D_{ci,i} = \sum_j R_{j,i}$.

Similarly to Pseudo Relevance Feedback, if we ignore the relation between documents and set $\beta = 0$, the ranking model degenerates to a linear ranking model in conventional local ranking.

### 4.2.2 Learning

In learning, we use Gradient Ascent to maximize the log likelihood. We use the same technique as that for PRF to guarantee $\alpha_k > 0$. The gradient of $L(\alpha, \beta)$ with respect to $\log \alpha_k$ and $\beta$ can be found[6] in Eq. (19) and (20). Due to space limitation, we omit the details of the learning algorithm, which is similar to Algorithm 1.

$$\nabla_{\log \alpha_k} = \frac{\partial L(\alpha, \beta)}{\partial \log \alpha_k} = \alpha_k \left\{ \frac{n}{2a} + \frac{1}{4a^2} b^T b - \frac{1}{2a} b^T X_{,k} + \sum_i x_{i,k}^2 - \sum_i (y_i - x_{i,k})^2 \right\} \tag{19}$$

$$\nabla_\beta = \frac{\partial L(\alpha, \beta)}{\partial \beta} = -\frac{1}{2a} b^T (D_r - D_c)e + \sum_{i,j} R_{i,j} (y_i - y_j) \tag{20}$$

where where $n$ denotes number of documents for the query, and $a = \alpha^T e$, $b = 2X\alpha + \beta(D_r - D_c)e$, $c = \sum_i \sum_{k=1}^{K_1} \alpha_k x_{i,k}^2$, $X_{,k}$ denotes the $k$-th column of matrix $X$.

### 4.3 Continuous CRF for Multiple Relations

We only consider using one type of relation in the previous two cases. We can also conduct global ranking by utilizing multiple types of relation. C-CRF is a powerful tool to perform the task. It can easily incorporate various types of relation as edge feature functions. For example, we can combine similarity relation and parent-child relation by using the following C-CRF model:

$$\Pr(y|x) = \frac{1}{Z(x)} \exp \left\{ \sum_i \sum_{k=1}^{K_1} -\alpha_k (y_i - x_{i,k})^2 + \sum_{i,j} \left( \beta_1 R_{i,j} (y_i - y_j) - \beta_2 \frac{S_{i,j}}{2} (y_i - y_j)^2 \right) \right\}.$$

In this case, the ranking scores of documents for a new query is calculated as follows.

$$F(x) = \arg \max_y \Pr(y|x; \alpha, \beta) = (\alpha^T eI + \beta_2 D - \beta_2 S)^{-1} \left( X\alpha + \frac{\beta_1}{2} (D_r - D_c)e \right)$$

## 5 Experiments

We empirically tested the performance of C-CRF on both Pseudo Relevance Feedback and Topic Distillation[7]. As data, we used LETOR [10], which is a public dataset for learning to rank research.

Table 1: Ranking Accuracy

| PRF on OHSUMED Data | | | | | TD on TREC2004 Data | | | |
|---|---|---|---|---|---|---|---|---|
| **Algorithms** | **ndcg1** | **ndcg2** | **ndcg5** | | **Algorithms** | **ndcg1** | **ndcg2** | **ndcg5** |
| BM25 | 0.3994 | 0.3931 | 0.3972 | | BM25 | 0.3067 | 0.2933 | 0.2293 |
| BM25-PRF | 0.3962 | 0.4277 | 0.3981 | | ST | 0.3200 | 0.3133 | 0.3232 |
| RankSVM | 0.4952 | 0.4755 | 0.4579 | | SS | 0.3200 | 0.3200 | 0.3227 |
| ListNet | 0.5231 | 0.497 | 0.4662 | | RankSVM | 0.4400 | 0.4333 | 0.3935 |
| C-CRF | **0.5443** | **0.4986** | **0.4808** | | ListNet | 0.4400 | 0.4267 | 0.4209 |
| | | | | | C-CRF | **0.5200** | **0.4733** | **0.4428** |

We made use of OHSUMED in LETOR for Pseudo Relevance Feedback and TREC2004 in LETOR for Topic Distillation. As evaluation measure, we utilized NDCG@n (Normalized Discounted Cumulative Gain) [6].

As baseline methods for the two tasks, we used several local ranking algorithms such as BM25, RankSVM [7] and ListNet [2]. BM25 is a widely used non-learning ranking method. RankSVM is a state-of-the-art algorithm of the pairwise approach to learning to rank, and ListNet is a state-of-the-art algorithm of the listwise approach. For Pseudo Relevance Feedback, we also compared with a traditional feedback method based on BM25 (BM25-PRF for short). For Topic Distillation, we also compared with two traditional methods, sitemap based term propagation (ST) and sitemap based score propagation (SS) [11], which propagate the relevance along sitemap structure. These algorithms can be regarded as a kind of global ranking methods but they are not based on supervised learning. We conducted 5 fold cross validation for C-CRF and all the baseline methods, using the partition provided in LETOR.

The left part of Table 1 shows the ranking accuracies of BM25, BM25-PRF, RankSVM, ListNet, and C-CRF, in terms of NDCG averaged over five trials on OHSUMED data. C-CRF's performance is superior to the performances of RankSVM and ListNet. This is particularly true for NDCG@1; C-CRF achieves about 5 points higher accuracy than RankSVM and more than 2 points higher accuracy than ListNet. The results indicate that C-CRF based global ranking can indeed improve search relevance. C-CRF also outperforms BM25-PRF, the traditional method of using similarity information for ranking. The result suggests that it is better to employ a supervised learning approach for the task.

The right part of Table 1 shows the performances of BM25, SS, ST, RankSVM, ListNet, and C-CRF model in terms of NDCG averaged over 5 trials on TREC data. C-CRF outperforms RankSVM and ListNet at all NDCG positions. This is particularly true for NDCG@1. C-CRF achieves 8 points higher accuracy than RankSVM and ListNet, which is a more than 15% relative improvement. The result indicates that C-CRF based global ranking can achieve better results than local ranking for this task. C-CRF also outperforms SS and ST, the traditional method of using parent-child information for Topic Distillation. The result suggests that it is better to employ a learning based approach.

# 6   Related Work

Most existing work on using relation information in learning is for classification (e.g., [19, 1]) and clustering (e.g., [4, 15]). To the best of our knowledge, there was not much work on using relation for ranking, except Relational Ranking SVM (RRSVM) proposed in [14], which is based on a similar motivation as our work.

There are large differences between RRSVM and C-CRF, however. For RRSVM, it is hard to combine the uses of multiple types of relation. In contrast, C-CRF can easily do it by incorporating the relations in different edge feature functions. There is a hyper parameter $\beta$ in RRSVM representing the trade-off between content and relation information. It needs to be manually tuned. This is not necessary for C-CRF, however, because the trade-off between them is handled naturally by the feature weights in the model, which can be learnt automatically. Furthermore, in some cases certain approximation must be made on the model in RRSVM (e.g. for Topic Distillation) in order to fit into the learning framework of SVM. Such kind of approximation is unnecessary in C-CRF anyway.

Besides, C-CRF achieves better ranking accuracy than that reported for RRSVM [14] on the same benchmark dataset.

## 7 Conclusions

We studied learning to rank methods for global ranking problem, in which we use both content information of objects and relation information between objects for ranking. A Continuous CRF (C-CRF) model was proposed for performing the learning task. Taking Pseudo Relevance Feedback and Topic Distillation as examples, we showed how to use C-CRF in global ranking. Experimental results on benchmark data show that C-CRF improves upon the baseline methods in the global ranking tasks.

There are still issues which we need to investigate at the next step. (1) We have studied the method of learning C-CRF with Maximum Likelihood Estimation. It is interesting to see how to apply Maximum A Posteriori Estimation to the problem. (2) We have assumed absolute ranking scores given in training data. We will study how to train C-CRF with relative preference data. (3) We have studied two global ranking tasks: Pseudo Relevance Feedback and Topic Distillation. We plan to look at other tasks in the future.

## Footnotes

[1] We omit superscript $(q)$ in this section when there is no confusion.

[2] Note that $S_{i,j}$ is *not* computed from the ranking factors of documents $x_i$ and $x_j$ but from their raw terms. For more details, please refer to our technique report [13].

[3] $\alpha_k > 0$ means that the factor $x_{i,k}$ is positively correlated with the ranking score $y_i$. Considering that some factor may be negatively correlated with $y_i$, we double a factor $x_{i,k}$ into two factors $x_{i,k}$ and $x_{i,k'} = -x_{i,k}$ in experiments. Then if $\alpha_{k'} > \alpha_k$, one can get the factor $x_{i,k}$ is negatively correlated with the ranking score $y_i$.

[4]Stochastic Gradient means conducting gradient ascent from one query to another.

[5]Details can be found in [13].

[6] Please refer to [13] for the derivation of the two equations.

[7] Please refer to [13] for more details of experiments.

## References

[1] M. Belkin, P. Niyogi, and V. Sindhwani. Manifold regularization: A geometric framework for learning from labeled and unlabeled examples. *J. Mach. Learn. Res.*, 7:2399–2434, 2006.

[2] Z. Cao, T. Qin, T.-Y. Liu, M.-F. Tsai, and H. Li. Learning to rank: from pairwise approach to listwise approach. In *ICML '07*, pages 129–136, 2007.

[3] W. Chu and Z. Ghahramani. Gaussian processes for ordinal regression. *Journal of Machine Learning Research*, 6:1019–1041, 2005.

[4] I. S. Dhillon. Co-clustering documents and words using bipartite spectral graph partitioning. In *KDD '01*.

[5] G. H. Golub and C. F. V. Loan. *Matrix computations (3rd ed.)*. Johns Hopkins University Press, 1996.

[6] K. Järvelin and J. Kekäläinen. Cumulated gain-based evaluation of ir techniques. *ACM Trans. Inf. Syst.*, 20(4):422–446, 2002.

[7] T. Joachims. Optimizing search engines using clickthrough data. In *KDD '02*, pages 133–142, 2002.

[8] K. L. Kwok. A document-document similarity measure based on cited titles and probability theory, and its application to relevance feedback retrieval. In *SIGIR '84*, pages 221–231, 1984.

[9] J. G. Lewis. Algorithm 582: The gibbs-poole-stockmeyer and gibbs-king algorithms for reordering sparse matrices. *ACM Trans. Math. Softw.*, 8(2):190–194, 1982.

[10] T.-Y. Liu, J. Xu, T. Qin, W.-Y. Xiong, and H. Li. Letor: Benchmark dataset for research on learning to rank for information retrieval. In *SIGIR '07 Workshop*, 2007.

[11] T. Qin, T.-Y. Liu, X.-D. Zhang, Z. Chen, and W.-Y. Ma. A study of relevance propagation for web search. In *SIGIR '05*, pages 408–415, 2005.

[12] T. Qin, T.-Y. Liu, X.-D. Zhang, G. Feng, D.-S. Wang, and W.-Y. Ma. Topic distillation via sub-site retrieval. *Information Processing & Management*, 43(2):445–460, 2007.

[13] T. Qin, T.-Y. Liu, X.-D. Zhang, D.-S. Wang, and H. Li. Global ranking of documents using continuous conditional random fields. Technical Report MSR-TR-2008-156, Microsoft Corporation, 2008.

[14] T. Qin, T.-Y. Liu, X.-D. Zhang, D.-S. Wang, W.-Y. Xiong, and H. Li. Learning to rank relational objects and its application to web search. In *WWW '08*, 2008.

[15] J. Shi and J. Malik. Normalized cuts and image segmentation. *IEEE Transactions on Pattern Analysis and Machine Intelligence*, 22(8):888–905, 2000.

[16] C. Sutton and A. McCallum. An introduction to conditional random fields for relational learning. In L. Getoor and B. Taskar, editors, *Introduction to Statistical Relational Learning*. MIT Press, 2006.

[17] T. Tao and C. Zhai. Regularized estimation of mixture models for robust pseudo-relevance feedback. In *SIGIR '06*, pages 162–169, 2006.

[18] C. X. Zhai, W. W. Cohen, and J. Lafferty. Beyond independent relevance: methods and evaluation metrics for subtopic retrieval. In *SIGIR '03*, pages 10–17, 2003.

[19] D. Zhou, O. Bousquet, T. Lal, J. Weston, and B. Schölkopf. Learning with local and global consistency, 2003. In 18th Annual Conf. on Neural Information Processing Systems.

